# QUIC & DIRTY: A Quadratic Approximation Approach for Dirty Statistical Models

**Cho-Jui Hsieh, Inderjit S. Dhillon, Pradeep Ravikumar**
University of Texas at Austin
Austin, TX 78712 USA
`{cjhsieh,inderjit,pradeepr}@cs.utexas.edu`

**Stephen Becker**
University of Colorado at Boulder
Boulder, CO 80309 USA
`stephen.becker@colorado.edu`

**Peder A. Olsen**
IBM T.J. Watson Research Center
Yorktown Heights, NY 10598 USA
`pederao@us.ibm.com`

## Abstract

In this paper, we develop a family of algorithms for optimizing "superposition-structured" or "dirty" statistical estimators for high-dimensional problems involving the minimization of the sum of a smooth loss function with a hybrid regularization. Most of the current approaches are first-order methods, including proximal gradient or Alternating Direction Method of Multipliers (ADMM). We propose a new family of second-order methods where we approximate the loss function using quadratic approximation. The superposition structured regularizer then leads to a subproblem that can be efficiently solved by alternating minimization. We propose a general active subspace selection approach to speed up the solver by utilizing the low-dimensional structure given by the regularizers, and provide convergence guarantees for our algorithm. Empirically, we show that our approach is more than 10 times faster than state-of-the-art first-order approaches for the latent variable graphical model selection problems and multi-task learning problems when there is more than one regularizer. For these problems, our approach appears to be the first algorithm that can extend active subspace ideas to multiple regularizers.

## 1 Introduction

From the considerable amount of recent research on high-dimensional statistical estimation, it has now become well understood that it is vital to impose structural constraints upon the statistical model parameters for their statistically consistent estimation. These structural constraints take the form of sparsity, group-sparsity, and low-rank structure, among others; see [18] for unified statistical views of such structural constraints. In recent years, such "clean" structural constraints are frequently proving insufficient, and accordingly there has been a line of work on "superposition-structured" or "dirty model" constraints, where the model parameter is expressed as the sum of a number of parameter components, each of which have their own structure. For instance, [4, 6] consider the estimation of a matrix that is neither low-rank nor sparse, but which can be decomposed into the sum of a low-rank matrix and a sparse outlier matrix (this corresponds to robust PCA when the matrix-structured parameter corresponds to a covariance matrix). [5] use such matrix decomposition to estimate the structure of latent-variable Gaussian graphical models. [15] in turn use a superposition of sparse and group-sparse structure for multi-task learning. For other recent work on such superposition-structured models, see [1, 7, 14]. For a unified statistical view of such superposition-structured models, and the resulting classes of $M$-estimators, please see [27].

Consider a general superposition-structured parameter $\bar{\boldsymbol{\theta}} := \sum_{r=1}^{k} \boldsymbol{\theta}^{(r)}$, where $\{\boldsymbol{\theta}^{(r)}\}_{r=1}^{k}$ are the parameter-components, each with their own structure. Let $\{\mathcal{R}^{(r)}(\cdot)\}_{r=1}^{k}$ be regularization functions suited to the respective parameter components, and let $\mathcal{L}(\cdot)$ be a (typically non-linear) loss function

that measures the goodness of fit of the superposition-structured parameter $\bar{\boldsymbol{\theta}}$ to the data. We now have the notation to consider a popular class of $M$-estimators studied in the papers above for these superposition-structured models:

$$\min_{\{\boldsymbol{\theta}^{(r)}\}_{r=1}^k} \left\{ \mathcal{L}\left( \sum_r \boldsymbol{\theta}^{(r)} \right) + \sum_r \lambda_r \mathcal{R}^{(r)}(\boldsymbol{\theta}^{(r)}) \right\} := F(\boldsymbol{\theta}), \tag{1}$$

where $\{\lambda_r\}_{r=1}^k$ are regularization penalties. In (1), the overall regularization contribution is separable in the individual parameter components, but the loss function term itself is not, and depends on the sum $\bar{\boldsymbol{\theta}} := \sum_{r=1}^k \boldsymbol{\theta}^{(r)}$. Throughout the paper, we use $\bar{\boldsymbol{\theta}}$ to denote the overall superposition-structured parameter, and $\boldsymbol{\theta} = [\boldsymbol{\theta}^{(1)}, \ldots, \boldsymbol{\theta}^{(k)}]$ to denote the concatenation of all the parameters.

Due to the wide applicability of this class of $M$-estimators in (1), there has been a line of work on developing efficient optimization methods for solving special instances of this class of $M$-estimators [14, 26], in addition to the papers listed above. In particular, due to the superposition-structure in (1) and the high-dimensionality of the problem, this class seems naturally amenable to a proximal gradient descent approach or the ADMM method [2, 17]; note that these are first-order methods and are thus very scalable.

In this paper, we consider instead a proximal Newton framework to minimize the $M$-estimation objective in (1). Specifically, we use iterative quadratic approximations, and for each of the quadratic subproblems, we use an alternating minimization approach to individually update each of the parameter components comprising the superposition-structure. Note that the Hessian of the loss might be structured, as for instance with the logdet loss for inverse covariance estimation and the logistic loss, which allows us to develop very efficient second-order methods. Even given this structure, solving the regularized quadratic problem in order to obtain the proximal Newton direction is too expensive due to the high dimensional setting. The key **algorithmic contribution** of this paper is in developing a general active subspace selection framework for general decomposable norms, which allows us to solve the proximal Newton steps over a significantly reduced search space. We are able to do so by leveraging the structural properties of decomposable regularization functions in the $M$-estimator in (1).

Our other key contribution is **theoretical**. While recent works [16, 21] have analyzed the convergence of proximal Newton methods, the superposition-structure here poses a key caveat: since the loss function term only depends on the sum of the individual parameter components, the Hessian is not positive-definite, as is required in previous analyses of proximal Newton methods. The theoretical analysis [9] relaxes this assumption by instead assuming the loss is self-concordant but again allows at most one regularizer. Another key theoretical difficulty is our use of active subspace selection, where we do not solve for the vanilla proximal Newton direction, but solve the proximal Newton step subproblem only over a *restricted subspace*, which moreover varies with each step. We deal with these issues and show **super-linear convergence** of the algorithm when the sub-problems are solved exactly. We apply our algorithm to two real world applications: latent Gaussian Markov random field (GMRF) structure learning (with low-rank + sparse structure), and multitask learning (with sparse + group sparse structure), and demonstrate that our algorithm is more than **ten** times faster than state-of-the-art methods.

Overall, our algorithmic and theoretical developments open up the state of the art but forbidding class of $M$-estimators in (1) to very large-scale problems.

**Outline of the paper.** We begin by introducing some background in Section 2. In Section 3, we propose our quadratic approximation framework with active subspace selection for general dirty statistical models. We derive the convergence guarantees of our algorithm in Section 4. Finally, in Section 5, we apply our model to solve two real applications, and show experimental comparisons with other state-of-the-art methods.

## 2   Background and Applications

**Decomposable norms.** We consider the case where all the regularizers $\{\mathcal{R}^{(r)}\}_{r=1}^k$ are decomposable norms $\|\cdot\|_{\mathcal{A}_r}$. A norm $\|\cdot\|$ is decomposable at $\boldsymbol{x}$ if there is a subspace $\mathcal{T}$ and a vector $\boldsymbol{e} \in \mathcal{T}$ such that the sub differential at $\boldsymbol{x}$ has the following form:

$$\partial\|\boldsymbol{x}\|_r = \{\boldsymbol{\rho} \in \mathbb{R}^n \mid \Pi_{\mathcal{T}}(\boldsymbol{\rho}) = \boldsymbol{e} \text{ and } \|\Pi_{\mathcal{T}^\perp}(\boldsymbol{\rho})\|_{\mathcal{A}_r}^* \leq 1\}, \tag{2}$$

where $\Pi_{\mathcal{T}}(\cdot)$ is the orthogonal projection onto $\mathcal{T}$, and $\|\boldsymbol{x}\|^* := \sup_{\|\boldsymbol{a}\| \leq 1} \langle \boldsymbol{x}, \boldsymbol{a} \rangle$ is the dual norm of $\|\cdot\|$. The decomposable norm was defined in [3, 18], and many interesting regularizers belong to this category, including:

- Sparse vectors: for the $\ell_1$ regularizer, $\mathcal{T}$ is the span of all points with the same support as $\boldsymbol{x}$.
- Group sparse vectors: suppose that the index set can be partitioned into a set of $N_{\mathcal{G}}$ disjoint groups, say $\mathcal{G} = \{G_1, \ldots, G_{N_{\mathcal{G}}}\}$, and define the $(1,\alpha)$-group norm by $\|\boldsymbol{x}\|_{1,\alpha} := \sum_{t=1}^{N_{\mathcal{G}}} \|\boldsymbol{x}_{G_t}\|_\alpha$. If $S_G$ denotes the subset of groups where $\boldsymbol{x}_{G_t} \neq 0$, then the subgradient has the following form:

$$\partial \|\boldsymbol{x}\|_{1,\alpha} := \{\boldsymbol{\rho} \mid \boldsymbol{\rho} = \sum_{t \in S_G} \boldsymbol{x}_{G_t}/\|\boldsymbol{x}_{G_t}\|_\alpha^* + \sum_{t \notin S_G} \boldsymbol{m}_t\},$$

where $\|\boldsymbol{m}_t\|_\alpha^* \leq 1$ for all $t \notin S_G$. Therefore, the group sparse norm is also decomposable with

$$\mathcal{T} := \{\boldsymbol{x} \mid \boldsymbol{x}_{G_t} = 0 \text{ for all } t \notin S_G\}. \tag{3}$$

- Low-rank matrices: for the nuclear norm regularizer $\|\cdot\|_*$, which is defined to be the sum of singular values, the subgradient can be written as

$$\partial \|X\|_* = \{UV^T + W \mid U^TW = 0, WV = 0, \|W\|_2 \leq 1\},$$

where $\|\cdot\|_2$ is the matrix 2 norm and $U, V$ are the left/right singular vectors of $X$ corresponding to *non-zero* singular values. The above subgradient can also be written in the decomposable form (2), where $\mathcal{T}$ is defined to be $\text{span}(\{\boldsymbol{u}_i \boldsymbol{v}_j^T\}_{i,j=1}^k)$ where $\{\boldsymbol{u}_i\}_{i=1}^k, \{\boldsymbol{v}_i\}_{i=1}^k$ are the columns of $U$ and $V$.

**Applications.** Next we discuss some widely used applications of superposition-structured models, and the corresponding instances of the class of $M$-estimators in (1).

- Gaussian graphical model with latent variables: let $\Theta$ denote the precision matrix with corresponding covariance matrix $\Sigma = \Theta^{-1}$. [5] showed that the precision matrix will have a low rank + sparse structure when some random variables are hidden, thus $\Theta = S - L$ can be estimated by solving the following regularized MLE problem:

$$\min_{S,L:L \succeq 0, S-L \succ 0} -\log\det(S - L) + \langle S - L, \Sigma \rangle + \lambda_S \|S\|_1 + \lambda_L \text{trace}(L). \tag{4}$$

While proximal Newton methods have recently become a dominant technique for solving the $\ell_1$-regularized log-determinant problems [12, 10, 13, 19], our development is the first to apply proximal Newton methods to solve log-determinant problems with sparse *and* low rank regularizers.

- Multi-task learning: given $k$ tasks, each with sample matrix $X^{(r)} \in \mathbb{R}^{n_r \times d}$ ($n_r$ samples in the $r$-th task) and labels $y^{(r)}$, [15] proposes minimizing the following objective:

$$\sum_{r=1}^k \ell(y^{(r)}, X^{(r)}(S^{(r)} + B^{(r)})) + \lambda_S \|S\|_1 + \lambda_B \|B\|_{1,\infty}, \tag{5}$$

where $\ell(\cdot)$ is the loss function and $S^{(r)}$ is the $r$-th column of $S$.

- Noisy PCA: to recover a covariance matrix corrupted with sparse noise, a widely used technique is to solve the matrix decomposition problem [6]. In contrast to the squared loss above, an exponential PCA problem [8] would use a Bregman divergence for the loss function.

## 3 Our proposed framework

To perform a Newton-like step, we iteratively form quadratic approximations of the smooth loss function. Generally the quadratic subproblem will have a large number of variables and will be hard to solve. Therefore we propose a general active subspace selection technique to reduce the problem size by exploiting the structure of the regularizers $\mathcal{R}_1, \ldots, \mathcal{R}_k$.

### 3.1 Quadratic Approximation

Given $k$ sets of variables $\boldsymbol{\theta} = [\boldsymbol{\theta}^{(1)}, \ldots, \boldsymbol{\theta}^{(k)}]$, and each $\boldsymbol{\theta}^{(r)} \in \mathbb{R}^n$, let $\boldsymbol{\Delta}^{(r)}$ denote perturbation of $\boldsymbol{\theta}^{(r)}$, and $\boldsymbol{\Delta} = [\boldsymbol{\Delta}^{(1)}, \ldots, \boldsymbol{\Delta}^{(k)}]$. We define $g(\boldsymbol{\theta}) := \mathcal{L}(\sum_{r=1}^k \boldsymbol{\theta}^{(r)}) = \mathcal{L}(\bar{\boldsymbol{\theta}})$ to be the loss function, and $h(\boldsymbol{\theta}) := \sum_{r=1}^k \mathcal{R}^{(r)}(\boldsymbol{\theta}^{(r)})$ to be the regularization. Given the current estimate $\boldsymbol{\theta}$, we form the quadratic approximation of the smooth loss function:

$$\bar{g}(\boldsymbol{\theta} + \boldsymbol{\Delta}) = g(\boldsymbol{\theta}) + \sum_{r=1}^k \langle \boldsymbol{\Delta}^{(r)}, G \rangle + \frac{1}{2}\boldsymbol{\Delta}^T \mathcal{H}\boldsymbol{\Delta}, \tag{6}$$

where $G = \nabla\mathcal{L}(\bar{\boldsymbol{\theta}})$ is the gradient of $\mathcal{L}$ and $\mathcal{H}$ is the Hessian matrix of $g(\boldsymbol{\theta})$. Note that $\nabla_{\bar{\boldsymbol{\theta}}}\mathcal{L}(\bar{\boldsymbol{\theta}}) = \nabla_{\boldsymbol{\theta}^{(r)}}\mathcal{L}(\bar{\boldsymbol{\theta}})$ for all $r$ so we simply write $\nabla$ and refer to the gradient at $\bar{\boldsymbol{\theta}}$ as $G$ (and similarly for $\nabla^2$). By the chain rule, we can show that

**Lemma 1.** *The Hessian matrix of $g(\boldsymbol{\theta})$ is*

$$\mathcal{H} := \nabla^2 g(\boldsymbol{\theta}) = \begin{bmatrix} H & \cdots & H \\ \vdots & \ddots & \vdots \\ H & \cdots & H \end{bmatrix}, \quad H := \nabla^2 \mathcal{L}(\bar{\boldsymbol{\theta}}). \quad (7)$$

In this paper we focus on the case where $H$ is positive definite. When it is not, we add a small constant $\epsilon$ to the diagonal of $H$ to ensure that each block is positive definite.

Note that the full Hessian, $\mathcal{H}$, will in general, *not* be positive definite (in fact $\text{rank}(\mathcal{H}) = \text{rank}(H)$). However, based on its special structure, we can still give convergence guarantees (along with rate of convergence) for our algorithm. The Newton direction $\boldsymbol{d}$ is defined to be:

$$[\boldsymbol{d}^{(1)}, \dots, \boldsymbol{d}^{(k)}] = \underset{\boldsymbol{\Delta}^{(1)}, \dots, \boldsymbol{\Delta}^{(k)}}{\text{argmin}} \; \bar{g}(\boldsymbol{\theta} + \boldsymbol{\Delta}) + \sum_{r=1}^{k} \lambda_r \|\boldsymbol{\theta}^{(r)} + \boldsymbol{\Delta}^{(r)}\|_{\mathcal{A}_r} := Q_{\mathcal{H}}(\boldsymbol{\Delta}; \boldsymbol{\theta}). \quad (8)$$

The quadratic subproblem (8) cannot be directly separated into $k$ parts because the Hessian matrix (7) is not a block-diagonal matrix. Also, each set of parameters has its own regularizer, so it is hard to solve them all together. Therefore, to solve (8), we propose a block coordinate descent method. At each iteration, we pick a variable set $\boldsymbol{\Delta}^{(r)}$ where $r \in \{1, 2, \dots, k\}$ by a cyclic (or random) order, and update the parameter set $\boldsymbol{\Delta}^{(r)}$ while keeping other parameters fixed. Assume $\boldsymbol{\Delta}$ is the current solution (for all the variable sets), then the subproblem with respect to $\boldsymbol{\Delta}^{(r)}$ can be written as

$$\boldsymbol{\Delta}^{(r)} \leftarrow \underset{\boldsymbol{d} \in \mathbb{R}^n}{\text{argmin}} \; \frac{1}{2} \boldsymbol{d}^T H \boldsymbol{d} + \langle \boldsymbol{d}, G + \sum_{t: r \neq t} H \boldsymbol{\Delta}^{(t)} \rangle + \lambda_r \|\boldsymbol{\theta}^{(r)} + \boldsymbol{d}\|_{\mathcal{A}_r}. \quad (9)$$

The subproblem (9) is just a typical quadratic problem with a specific regularizer, so there already exist efficient algorithms for solving it for different choices of $\| \cdot \|_{\mathcal{A}}$. For the $\ell_1$ norm regularizer, coordinate descent methods can be applied to solve (9) efficiently as used in [12, 21]; (accelerated) proximal gradient descent or projected Newton's method can also be used, as shown in [19]. For a general atomic norm where there might be infinitely many atoms (coordinates), a greedy coordinate descent approach can be applied, as shown in [22].

To iterate between different groups of parameters, we have to maintain the term $\sum_{r=1}^{k} H \boldsymbol{\Delta}^{(r)}$ during the Newton iteration. Directly computing $H \boldsymbol{\Delta}^{(r)}$ requires $O(n^2)$ flops; however, the Hessian matrix often has a special structure so that $H \boldsymbol{\Delta}^{(r)}$ can be computed efficiently. For example, in the inverse covariance estimation problem $H = \Theta^{-1} \otimes \Theta^{-1}$ where $\Theta^{-1}$ is the current estimate of covariance, and in the empirical risk minimization problem $H = X D X^T$ where $X$ is the data matrix and $D$ is diagonal.

After solving the subproblem (8), we have to search for a suitable stepsize. We apply an Armijo rule for line search [24], where we test the step size $\alpha = 2^0, 2^{-1}, \dots$ until the following sufficient decrease condition is satisfied for a pre-specified $\sigma \in (0, 1)$ (typically $\sigma = 10^{-4}$):

$$F(\boldsymbol{\theta} + \alpha \boldsymbol{\Delta}) \leq F(\boldsymbol{\theta}) + \alpha \sigma \delta, \quad \delta = \langle G, \boldsymbol{\Delta} \rangle + \sum_{r=1}^{k} \lambda_r \|\Theta_r + \alpha \boldsymbol{\Delta}^{(r)}\|_{\mathcal{A}_r} - \sum_{r=1}^{k} \lambda_r \|\boldsymbol{\theta}^{(r)}\|_{\mathcal{A}_r}. \quad (10)$$

### 3.2 Active Subspace Selection

Since the quadratic subproblem (8) contains a large number of variables, directly applying the above quadratic approximation framework is not efficient. In this subsection, we provide a general *active subspace selection* technique, which dramatically reduces the size of variables by exploiting the structure of regularizers. A similar method has been discussed in [12] for the $\ell_1$ norm and in [11] for the nuclear norm, but it has not been generalized to all decomposable norms. Furthermore, a key point to note is that in this paper our active subspace selection is not only a heuristic, but comes with strong convergence guarantees that we derive in Section 4.

Given the current $\boldsymbol{\theta}$, our subspace selection approach partitions each $\boldsymbol{\theta}^{(r)}$ into $\mathcal{S}_{\text{fixed}}^{(r)}$ and $\mathcal{S}_{\text{free}}^{(r)} = (\mathcal{S}_{\text{fixed}}^{(r)})^{\perp}$ and then restricts the search space of the Newton direction in (8) within $\mathcal{S}_{\text{free}}^{(r)}$, which yields the following quadratic approximation problem:

$$[\boldsymbol{d}^{(1)}, \dots, \boldsymbol{d}^{(k)}] = \underset{\boldsymbol{\Delta}^{(1)} \in \mathcal{S}_{\text{free}}^{(1)}, \dots, \boldsymbol{\Delta}^{(k)} \in \mathcal{S}_{\text{free}}^{(k)}}{\text{argmin}} \; \bar{g}(\boldsymbol{\theta} + \boldsymbol{\Delta}) + \sum_{r=1}^{k} \lambda_r \|\boldsymbol{\theta}^{(r)} + \boldsymbol{\Delta}^{(r)}\|_{\mathcal{A}_r}. \quad (11)$$

Each group of parameter has its own fixed/free subspace, so we now focus on a single parameter component $\boldsymbol{\theta}^{(r)}$. An ideal subspace selection procedure would satisfy:

Property (I). Given the current iterate $\boldsymbol{\theta}$, any updates along directions in the fixed set, for instance as $\boldsymbol{\theta}^{(r)} \leftarrow \boldsymbol{\theta}^{(r)} + \boldsymbol{a}$, $\boldsymbol{a} \in \mathcal{S}_{\text{fixed}}^{(r)}$, does not improve the objective function value.

Property (II). The subspace $\mathcal{S}_{\text{free}}$ converges to the support of the final solution in a finite number of iterations.

Suppose given the current iterate, we first do updates along directions in the fixed set, and then do updates along directions in the free set. Property (I) ensures that this is equivalent to ignoring updates along directions in the fixed set in this current iteration, and focusing on updates along the free set. As we will show in the next section, this property would suffice to ensure global convergence of our procedure. Property (II) will be used to derive the asymptotic quadratic convergence rate.

We will now discuss our active subspace selection strategy which will satisfy both properties above. Consider the parameter component $\boldsymbol{\theta}^{(r)}$, and its corresponding regularizer $\| \cdot \|_{\mathcal{A}_r}$. Based on the definition of decomposable norm in (2), there exists a subspace $\mathcal{T}_r$ where $\Pi_{\mathcal{T}_r}(\boldsymbol{\rho})$ is a fixed vector for any subgradient of $\| \cdot \|_{\mathcal{A}_r}$. The following proposition explores some properties of the subdifferential of the overall objective $F(\boldsymbol{\theta})$ in (1).

**Proposition 1.** *Consider any unit-norm vector $\boldsymbol{a}$, with $\|\boldsymbol{a}\|_{\mathcal{A}_r} = 1$, such that $\boldsymbol{a} \in \mathcal{T}_r^{\perp}$.*

*(a) The inner-product of the sub-differential $\partial_{\boldsymbol{\theta}^{(r)}} F(\boldsymbol{\theta})$ with $\boldsymbol{a}$ satisfies:*

$$\langle \boldsymbol{a}, \partial_{\boldsymbol{\theta}^{(r)}} F(\boldsymbol{\theta}) \rangle \in [\langle \boldsymbol{a}, G \rangle - \lambda_r, \langle \boldsymbol{a}, G \rangle + \lambda_r]. \tag{12}$$

*(b) Suppose $|\langle \boldsymbol{a}, G \rangle| \le \lambda_r$. Then, $0 \in \operatorname{argmin}_{\sigma} F(\boldsymbol{\theta} + \sigma \boldsymbol{a})$.*

See Appendix 7.8 for the proof. Note that $G = \nabla \mathcal{L}(\bar{\boldsymbol{\theta}})$ denotes the gradient of $\mathcal{L}$. The proposition thus implies that if $|\langle \boldsymbol{a}, G \rangle| \le \lambda_r$ and $\mathcal{S}_{\text{fixed}}^{(r)} \subset \mathcal{T}_r^{\perp}$ then Property (I) immediately follows. The difficulty is that the set $\{\boldsymbol{a} \mid |\langle \boldsymbol{a}, G \rangle| \le \lambda_r\}$ is possibly hard to characterize, and even if we could characterize this set, it may not be amenable enough for the optimization solvers to leverage in order to provide a speedup. Therefore, we propose an alternative characterization of the fixed subspace:

**Definition 1.** *Let $\boldsymbol{\theta}^{(r)}$ be the current iterate, $\operatorname{prox}_{\lambda}^{(r)}$ be the proximal operator defined by*

$$\operatorname{prox}_{\lambda}^{(r)}(\boldsymbol{x}) = \operatorname*{argmin}_{\boldsymbol{y}} \frac{1}{2} \|\boldsymbol{y} - \boldsymbol{x}\|^2 + \lambda \|\boldsymbol{y}\|_{\mathcal{A}_r},$$

*and $\mathcal{T}_r(\boldsymbol{x})$ be the subspace for the decomposable norm (2) $\| \cdot \|_{\mathcal{A}_r}$ at point $\boldsymbol{x}$. We can define the fixed/free subset at $\boldsymbol{\theta}^{(r)}$ as:*

$$\mathcal{S}_{\text{fixed}}^{(r)} := [\mathcal{T}(\boldsymbol{\theta}^{(r)})]^{\perp} \cap [\mathcal{T}(\operatorname{prox}_{\lambda_r}^{(r)}(G))]^{\perp}, \quad \mathcal{S}_{\text{free}}^{(r)} = \mathcal{S}_{\text{fixed}}^{(r)}{}^{\perp}. \tag{13}$$

It can be shown that from the definition of the proximal operator, and Definition 1, it holds that $|\langle \boldsymbol{a}, G \rangle| < \lambda_r$, so that we would have local optimality in the direction $\boldsymbol{a}$ as before. We have the following proposition:

**Proposition 2.** *Let $\mathcal{S}_{\text{fixed}}^{(r)}$ be the fixed subspace defined in Definition 1. We then have:*

$$0 = \operatorname*{argmin}_{\boldsymbol{\Delta}^{(r)} \in \mathcal{S}_{\text{fixed}}^{(r)}} Q_{\mathcal{H}}([\boldsymbol{0}, \dots, \boldsymbol{0}, \boldsymbol{\Delta}^{(r)}, \boldsymbol{0}, \dots, \boldsymbol{0}]; \boldsymbol{\theta}).$$

We will prove that $\mathcal{S}_{\text{free}}$ as defined above converges to the final support in Section 4, as required in Property (II) above. We will now detail some examples of the fixed/free subsets defined above.

- For $\ell_1$ regularization: $\mathcal{S}_{\text{fixed}} = \operatorname{span}\{e_i \mid \theta_i = 0 \text{ and } |\nabla_i \mathcal{L}(\bar{\boldsymbol{\theta}})| \le \lambda\}$ where $e_i$ is the $i^{\text{th}}$ canonical vector.
- For nuclear norm regularization: the selection scheme can be written as

$$\mathcal{S}_{\text{free}} = \{U_A M V_A^T \mid M \in \mathbb{R}^{k \times k}\}, \tag{14}$$

where $U_A = \operatorname{span}(U, U_g)$, $V_A = \operatorname{span}(V, V_g)$, with $\Theta = U \Sigma V^T$ is the thin SVD of $\Theta$ and $U_g, V_g$ are the left and right singular vectors of $\operatorname{prox}_{\lambda}(\Theta - \nabla \mathcal{L}(\Theta))$. The proximal operator $\operatorname{prox}_{\lambda}(\cdot)$ in this case corresponds to singular-value soft-thresholding, and can be computed by randomized SVD or the Lanczos algorithm.

- For group sparse regularization: in the $(1, 2)$-group norm case, let $S_G$ be the nonzero groups, then the fixed groups $F_G$ can be defined by $F_G := \{i \mid i \notin S_G \text{ and } \|\nabla \mathcal{L}_{G_i}(\bar{\boldsymbol{\theta}})\| \leq \lambda\}$, and the free subspace will be

$$\mathcal{S}_{\text{free}} = \{\boldsymbol{\theta} \mid \boldsymbol{\theta}_i = 0 \; \forall i \in F_G\}. \tag{15}$$

In Figure 3 (in the appendix) that the active subspace selection can significantly improve the speed for the block coordinate descent algorithm [20].

---

**Algorithm 1:** QUIC & DIRTY: Quadratic Approximation Framework for Dirty Statistical Models

**Input** : Loss function $\mathcal{L}(\cdot)$, regularizers $\lambda_r \| \cdot \|_{\mathcal{A}_r}$ for $r = 1, \ldots, k$, and initial iterate $\boldsymbol{\theta}_0$.
**Output**: Sequence $\{\boldsymbol{\theta}_t\}$ such that $\{\bar{\boldsymbol{\theta}}_t\}$ converges to $\bar{\boldsymbol{\theta}}^\star$.

**1 for** $t = 0, 1, \ldots$ **do**
**2**      Compute $\bar{\boldsymbol{\theta}}_t \leftarrow \sum_{r=1}^{k} \boldsymbol{\theta}_t^{(r)}$.
**3**      Compute $\nabla \mathcal{L}(\bar{\boldsymbol{\theta}}_t)$.
**4**      Compute $\mathcal{S}_{\text{free}}$ by (13).
**5**      **for** $sweep = 1, \ldots, T_{outer}$ **do**
**6**          **for** $r = 1, \ldots, k$ **do**
**7**              Solve the subproblem (9) within $\mathcal{S}_{\text{free}}^{(r)}$.
**8**              Update $\sum_{r=1}^{k} \nabla^2 \mathcal{L}(\bar{\boldsymbol{\theta}}_t) \boldsymbol{\Delta}^{(r)}$.
**9**      Find the step size $\alpha$ by (10).
**10**     $\boldsymbol{\theta}^{(r)} \leftarrow \boldsymbol{\theta}^{(r)} + \alpha \boldsymbol{\Delta}^{(r)}$ for all $r = 1, \ldots, k$.

---

## 4 Convergence

The recently developed theoretical analysis of proximal Newton methods [16, 21] cannot be directly applied because (1) we have the active subspace selection step, and (2) the Hessian matrix for each quadratic subproblem is *not* positive definite. We first prove the global convergence of our algorithm when the quadratic approximation subproblem (11) is solved exactly. Interestingly, in our proof we show that the active subspace selection can be modeled within the framework of the Block Coordinate Gradient Descent algorithm [24] with a carefully designed Hessian approximation, and by making this connection we are able to prove global convergence.

**Theorem 1.** *Suppose $\mathcal{L}(\cdot)$ is convex (may not be strongly convex), and the quadratic subproblem (8) at each iteration is solved exactly, Algorithm 1 converges to the optimal solution.*

The proof is in Appendix 7.1. Next we consider the case that $\mathcal{L}(\bar{\boldsymbol{\theta}})$ is strongly convex. Note that even when $\mathcal{L}(\bar{\boldsymbol{\theta}})$ is strongly convex with respect to $\bar{\boldsymbol{\theta}}$, $\mathcal{L}(\sum_{r=1}^{k} \boldsymbol{\theta}^{(r)})$ will not be strongly convex in $\boldsymbol{\theta}$ (if $k > 1$) and there may exist more than one optimal solution. However, we show that all solutions give the same $\bar{\boldsymbol{\theta}} := \sum_{r=1}^{k} \boldsymbol{\theta}^{(r)}$.

**Lemma 2.** *Assume $\mathcal{L}(\cdot)$ is strongly convex, and $\{\boldsymbol{x}^{(r)}\}_{r=1}^{k}$, $\{\boldsymbol{y}^{(r)}\}_{r=1}^{k}$ are two optimal solutions of (1), then $\sum_{r=1}^{k} \boldsymbol{x}^{(r)} = \sum_{r=1}^{k} \boldsymbol{y}^{(r)}$.*

The proof is in Appendix 7.2. Next, we show that $\mathcal{S}_{\text{free}}^{(r)}$ (from Definition 1) will converge to the final support $\bar{\mathcal{T}}^{(r)}$ for each parameter set $r = 1, \ldots, k$. Let $\bar{\boldsymbol{\theta}}^\star$ be the global minimizer (which is unique as shown in Lemma 2), and assume that we have

$$\|\Pi_{(\bar{\mathcal{T}}^{(r)})^\perp} \left( \nabla \mathcal{L}(\bar{\boldsymbol{\theta}}^\star) \right) \|_{\mathcal{A}_r}^* < \lambda_r \; \forall r = 1, \ldots, k. \tag{16}$$

This is the generalization of the assumption used in earlier literature [12] where only $\ell_1$ regularization was considered. The condition is similar to strict complementary in linear programming.

**Theorem 2.** *If $\mathcal{L}(\cdot)$ is strongly convex and assumption (16) holds, then there exists a finite $T > 0$ such that $\mathcal{S}_{free}^{(r)} = \bar{\mathcal{T}}^{(r)} \;\; \forall r = 1, \ldots, k$ after $t > T$ iterations.*

The proof is in Appendix 7.3. Next we show that our algorithm has an asymptotic quadratic convergence rate (the proof is in Appendix 7.4).

**Theorem 3.** *Assume that $\nabla^2 \mathcal{L}(\cdot)$ is Lipschitz continuous, and assumption (16) holds. If at each iteration the quadratic subproblem (8) is solved exactly, and $\mathcal{L}(\cdot)$ is strongly convex, then our algorithm converges with asymptotic quadratic convergence rate.*

# 5 Applications

We demonstrate that our algorithm is extremely efficient for two applications: Gaussian Markov Random Fields (GMRF) with latent variables (with sparse + low rank structure) and multi-task learning problems (with sparse + group sparse structure).

## 5.1 GMRF with Latent Variables

We first apply our algorithm to solve the latent feature GMRF structure learning problem in eq (4), where $S \in \mathbb{R}^{p \times p}$ is the sparse part, $L \in \mathbb{R}^{p \times p}$ is the low-rank part, and we require $L = L^T \succeq 0, S = S^T$ and $Y = S - L \succ 0$ (i.e. $\boldsymbol{\theta}^{(2)} = -L$). In this case, $\mathcal{L}(Y) = -\log \det(Y) + \langle \Sigma, Y \rangle$, hence

$$\nabla^2 \mathcal{L}(Y) = Y^{-1} \otimes Y^{-1}, \text{ and } \nabla \mathcal{L}(Y) = \Sigma - Y^{-1}. \tag{17}$$

**Active Subspace.** For the sparse part, the free subspace is a subset of indices $\{(i,j) \mid S_{ij} \neq 0 \text{ or } |\nabla_{ij}\mathcal{L}(Y)| \geq \lambda\}$. For the low-rank part, the free subspace can be presented as $\{U_A M V_A^T \mid M \in \mathbb{R}^{k \times k}\}$ where $U_A$ and $V_A$ are defined in (14).

**Updating $\Delta_L$.** To solve the quadratic subproblem (11), first we discuss how to update $\Delta_L$ using subspace selection. The subproblem is

$$\min_{\Delta_L = U\Delta_D U^T : L + \Delta_L \succeq 0} \frac{1}{2} \operatorname{trace}(\Delta_L Y^{-1} \Delta_L Y^{-1}) + \operatorname{trace}((Y^{-1} - \Sigma - Y^{-1}\Delta_S Y^{-1})\Delta_L) + \lambda_L \|L + \Delta_L\|_*,$$

and since $\Delta_L$ is constrained to be a perturbation of $L = U_A M U_A^T$ so that we can write $\Delta_L = U_A \Delta_M U_A^T$, and the subproblem becomes

$$\min_{\Delta_M : M + \Delta_M \succeq 0} \frac{1}{2} \operatorname{trace}(\bar{Y}\Delta_M \bar{Y}\Delta_M) + \operatorname{trace}(\bar{\Sigma}\Delta_M) + \lambda_L \operatorname{trace}(M + \Delta_M) := q(\Delta_M), \tag{18}$$

where $\bar{Y} := U_A^T Y^{-1} U_A$ and $\bar{\Sigma} := U_A^T (Y^{-1} - \Sigma - Y^{-1}\Delta_S Y^{-1})U_A$. Therefore the subproblem (18) becomes a $k \times k$ dimensional problem where $k \ll p$.

To solve (18), we first check if the closed form solution exists. Note that $\nabla q(\Delta_M) = \bar{Y}\Delta_M \bar{Y} + \bar{\Sigma} + \lambda_L I$, thus the minimizer is $\Delta_M = -\bar{Y}^{-1}(\bar{\Sigma} + \lambda_L I)\bar{Y}^{-1}$ if $M + \Delta_M \succeq 0$. If not, we solve the subproblem by the projected gradient descent method, where each step only requires $O(k^2)$ time.

**Updating $\Delta_S$.** The subproblem with respect to $\Delta_S$ can be written as

$$\min_{\Delta_S} \frac{1}{2} \operatorname{vec}(\Delta_S)^T (Y^{-1} \otimes Y^{-1}) \operatorname{vec}(\Delta_S) + \operatorname{trace}((\Sigma - Y^{-1} - Y^{-1}(\Delta_L)Y^{-1})\Delta_S) + \lambda_S \|S + \Delta_S\|_1,$$

In our implementation we apply the same coordinate descent procedure proposed in QUIC [12] to solve this subproblem.

**Results.** We compare our algorithm with two state-of-the-art software packages. The LogdetPPA algorithm was proposed in [26] and used in [5] to solve (4). The PGALM algorithm was proposed in [17]. We run our algorithm on three gene expression datasets: the ER dataset ($p = 692$), the Leukemia dataset ($p = 1255$), and a subset of the Rosetta dataset ($p = 2000$)[1] For the parameters, we use $\lambda_S = 0.5, \lambda_L = 50$ for the ER and Leukemia datasets, which give us low-rank and sparse results. For the Rosetta dataset, we use the parameters suggested in LogdetPPA, with $\lambda_S = 0.0313, \lambda_L = 0.1565$. The results in Figure 1 shows that our algorithm is more than 10 times faster than other algorithms. Note that in the beginning PGALM tends to produce infeasible solutions ($L$ or $S - L$ is not positive definite), which is not plotted in the figures.

Our proximal Newton framework has two algorithmic components: the quadratic approximation, and our active subspace selection. From Figure 1 we can observe that although our algorithm is a Newton-like method, the time cost for each iteration is similar or even cheaper than other first order methods. The reason is (1) we take advantage from active selection, and (2) the problem has a special structure of the Hessian (17), where computing it is no more expensive than the gradient. To delineate the contribution of the quadratic approximation to the gain in speed of convergence, we further compare our algorithm to an alternating minimization approach for solving (4), together with our active subspace selection. Such an alternating minimization approach would iteratively fix one of $S, L$, and update the other; we defer detailed algorithmic and implementation details to Appendix 7.6 for reasons of space. The results show that by using the quadratic approximation, we get a much faster convergence rate (see Figure 2 in Appendix 7.6).

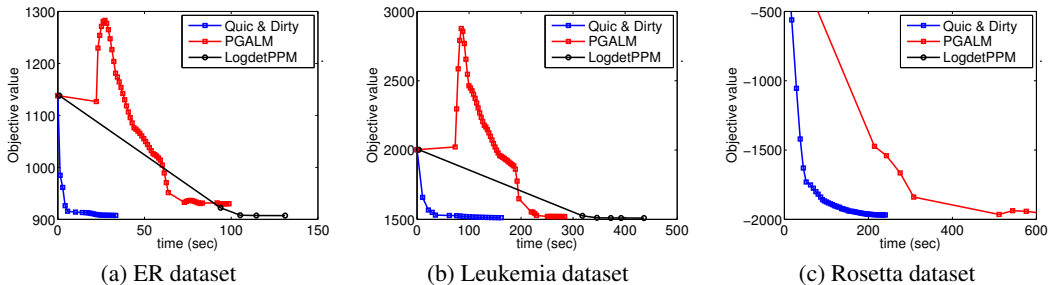

| (a) ER dataset | (b) Leukemia dataset | (c) Rosetta dataset |
| --- | --- | --- |

Figure 1: Comparison of algorithms on the latent feature GMRF problem using gene expression datasets. Our algorithm is much faster than PGALM and LogdetPPA.

Table 1: The comparisons on multi-task problems.

| dataset | number of training data | relative error | Dirty Models (sparse + group sparse) | | | Other Models | |
| --- | --- | --- | --- | --- | --- | --- | --- |
| | | | QUIC & DIRTY | proximal gradient | ADMM | Lasso | Group Lasso |
| USPS | 100 | $10^{-1}$ | 8.3% / **0.42s** | 8.5% / 1.8s | 8.3% / 1.3 | 10.27% | 8.36% |
| | 100 | $10^{-4}$ | 7.47% / **0.75s** | 7.49% / 10.8s | 7.47% / 4.5s | | |
| | 400 | $10^{-1}$ | 2.92% / **1.01s** | 2.9% / 9.4s | 3.0% / 3.6s | 4.87% | 2.93% |
| | 400 | $10^{-4}$ | 2.5% / **1.55s** | 2.5% / 35.8 | 2.5% / 11.0s | | |
| RCV1 | 1000 | $10^{-1}$ | 18.91% / **10.5s** | 18.5%/47s | 18.9% / 23.8s | 22.67% | 20.8% |
| | 1000 | $10^{-4}$ | 18.45% / **23.1s** | 18.49% / 430.8s | 18.5% / 259s | | |
| | 5000 | $10^{-1}$ | 10.54% / **42s** | 10.8% / 541s | 10.6% / 281s | 13.67% | 12.25% |
| | 5000 | $10^{-4}$ | 10.27% / **87s** | 10.27% / 2254s | 10.27% / 1191s | | |

## 5.2 Multiple-task learning with superposition-structured regularizers

Next we solve the multi-task learning problem (5) where the parameter is a sparse matrix $S \in \mathbb{R}^{d \times k}$ and a group sparse matrix $B \in \mathbb{R}^{d \times k}$. Instead of using the square loss (as in [15]), we consider the logistic loss $\ell_{\text{logistic}}(y, a) = \log(1 + e^{-ya})$, which gives better performance as seen by comparing Table 1 to results in [15]. Here the Hessian matrix has a special structure again: $H = XDX^T$ where $X$ is the data matrix and $D$ is the diagonal matrix, and in Appendix 7.7 we have a detail description of how to applying our algorithm to solve this problem.

**Results.** We follow [15] and transform multi-class problems into multi-task problems. For a multiclass dataset with $k$ classes and $n$ samples, for each $r = 1, \ldots, k$, we generate $\boldsymbol{y}^r \in \{0, 1\}^n$ to be the vector such that $y_i^{(k)} = 1$ if and only if the $i$-th sample is in class $r$. Our first dataset is the USPS dataset which was first collected in [25] and subsequently widely used in multi-task papers. On this dataset, the use of several regularizers is crucial for good performance. For example, [15] demonstrates that on USPS, using lasso and group lasso regularizations together outperforms models with a single regularizer. However, they only consider the squared loss in their paper, whereas we consider a logistic loss which leads to better performance. For example, we get 7.47% error rate using 100 samples in USPS dataset, while using the squared loss the error rate is 10.8% [15]. Our second dataset is a larger document dataset RCV1 downloaded from LIBSVM Data, which has 53 classes and 47,236 features. We show that our algorithm is much faster than other algorithms on both datasets, especially on RCV1 where we are more than 20 times faster than proximal gradient descent. Here our subspace selection techniques works well because we expect that the active subspace at the true solution is small.

## 6 Acknowledgements

This research was supported by NSF grants CCF-1320746 and CCF-1117055. C.-J.H also acknowledges support from an IBM PhD fellowship. P.R. acknowledges the support of ARO via W911NF-12-1-0390 and NSF via IIS-1149803, IIS-1447574, and DMS-1264033. S.R.B. was supported by an IBM Research Goldstine Postdoctoral Fellowship while the work was performed.

## Footnotes

[1]The full dataset has $p = 6316$ but the other methods cannot solve this size problem.

# References

[1] A. Agarwal, S. Negahban, and M. J. Wainwright. Noisy matrix decomposition via convex relaxation: Optimal rates in high dimensions. *Annals if Statistics*, 40(2):1171–1197, 2012.

[2] S. Boyd, N. Parikh, E. Chu, B. Peleato, and J. Eckstein. Distributed optimization and statistical learning via the alternating direction method of multipliers. *Foundations and Trends in Machine Learning*, 3(1):1–122, 2011.

[3] E. Candes and B. Recht. Simple bounds for recovering low-complexity models. *Mathemetical Programming*, 2012.

[4] E. J. Candes, X. Li, Y. Ma, and J. Wright. Robust principal component analysis? *J. Assoc. Comput. Mach.*, 58(3):1–37, 2011.

[5] V. Chandrasekaran, P. A. Parrilo, and A. S. Willsky. Latent variable graphical model selection via convex optimization. *The Annals of Statistics*, 2012.

[6] V. Chandrasekaran, S. Sanghavi, P. A. Parrilo, and A. S. Willsky. Rank-sparsity incoherence for matrix decomposition. *Siam J. Optim*, 21(2):572–596, 2011.

[7] Y. Chen, A. Jalali, S. Sanghavi, and C. Caramanis. Low-rank matrix recovery from errors and erasures. *IEEE Transactions on Information Theory*, 59(7):4324–4337, 2013.

[8] M. Collins, S. Dasgupta, and R. E. Schapire. A generalization of principal component analysis to the exponential family. In *NIPS*, 2012.

[9] Q. T. Dinh, A. Kyrillidis, and V. Cevher. An inexact proximal path-following algorithm for constrained convex minimization. *arxiv:1311.1756*, 2013.

[10] C.-J. Hsieh, I. S. Dhillon, P. Ravikumar, and A. Banerjee. A divide-and-conquer method for sparse inverse covariance estimation. In *NIPS*, 2012.

[11] C.-J. Hsieh and P. A. Olsen. Nuclear norm minimization via active subspace selection. In *ICML*, 2014.

[12] C.-J. Hsieh, M. A. Sustik, I. S. Dhillon, and P. Ravikumar. Sparse inverse covariance matrix estimation using quadratic approximation. In *NIPS*, 2011.

[13] C.-J. Hsieh, M. A. Sustik, I. S. Dhillon, P. Ravikumar, and R. A. Poldrack. BIG & QUIC: Sparse inverse covariance estimation for a million variables. In *NIPS*, 2013.

[14] D. Hsu, S. M. Kakade, and T. Zhang. Robust matrix decomposition with sparse corruptions. *IEEE Trans. Inform. Theory*, 57:7221–7234, 2011.

[15] A. Jalali, P. Ravikumar, S. Sanghavi, and C. Ruan. A dirty model for multi-task learning. In *NIPS*, 2010.

[16] J. D. Lee, Y. Sun, and M. A. Saunders. Proximal Newton-type methods for convex optimization. In *NIPS*, 2012.

[17] S. Ma, L. Xue, and H. Zou. Alternating direction methods for latent variable Gaussian graphical model selection. *Neural Computation*, 25(8):2172–2198, 2013.

[18] S. N. Negahban, P. Ravikumar, M. J. Wainwright, and B. Yu. A unified framework for high-dimensional analysis of m-estimators with decomposable regularizers. *Statistical Science*, 27(4):538–557, 2012.

[19] P. Olsen, F. Oztoprak, J. Nocedal, and S. Rennie. Newton-like methods for sparse inverse covariance estimation. In *NIPS*, 2012.

[20] Z. Qin, K. Scheinberg, and D. Goldfarb. Efficient block-coordinate descent algorithm for the group lasso. *Mathematical Programming Computation*, 2013.

[21] K. Scheinberg and X. Tang. Practical inexact proximal quasi-newton method with global complexity analysis. *arxiv:1311.6547*, 2014.

[22] A. Tewari, P. Ravikumar, and I. Dhillon. Greedy algorithms for structurally constrained high dimensional problems. In *NIPS*, 2011.

[23] K.-C. Toh, P. Tseng, and S. Yun. A block coordinate gradient descent method for regularized convex separable optimization and covariance selection. *Mathemetical Programming*, 129:331–355, 2011.

[24] P. Tseng and S. Yun. A coordinate gradient descent method for nonsmooth separable minimization. *Mathematical Programming*, 117:387–423, 2007.

[25] M. van Breukelen, R. P. W. Duin, D. M. J. Tax, and J. E. den Hartog. Handwritten digit recognition by combined classifiers. *Kybernetika*, 34(4):381–386, 1998.

[26] C. Wang, D. Sun, and K.-C. Toh. Solving log-determinant optimization problems by a Newton-CG primal proximal point algorithm. *SIAM J. Optimization*, 20:2994–3013, 2010.

[27] E. Yang and P. Ravikumar. Dirty statistical models. In *NIPS*, 2013.

[28] E.-H. Yen, C.-J. Hsieh, P. Ravikumar, and I. S. Dhillon. Constant nullspace strong convexity and fast convergence of proximal methods under high-dimensional settings. In *NIPS*, 2014.

[29] G.-X. Yuan, C.-H. Ho, and C.-J. Lin. An improved GLMNET for L1-regularized logistic regression. *JMLR*, 13:1999–2030, 2012.

